# Estimating Equivalent Kernels For Neural Networks: A Data Perturbation Approach

**A. Neil Burgess**

Department of Decision Science
London Business School
London, NW1 4SA, UK

(N.Burgess@lbs.lon.ac.uk)

## ABSTRACT

We describe the notion of "equivalent kernels" and suggest that this provides a framework for comparing different classes of regression models, including neural networks and both parametric and non-parametric statistical techniques. Unfortunately, standard techniques break down when faced with models, such as neural networks, in which there is more than one "layer" of adjustable parameters. We propose an algorithm which overcomes this limitation, estimating the equivalent kernels for neural network models using a data perturbation approach. Experimental results indicate that the networks do not use the maximum possible number of degrees of freedom, that these can be controlled using regularisation techniques and that the equivalent kernels learnt by the network vary both in "size" and in "shape" in different regions of the input space.

## 1    INTRODUCTION

The dominant approaches within the statistical community, such as multiple linear regression but even extending to advanced techniques such as generalised additive models (Hastie and Tibshirani, 1990), projection pursuit regression (Friedman and Stuetzle, 1981), and classification and regression trees (Breiman et al., 1984), tend to err, when they do, on the high-bias side due to restrictive assumptions regarding either the functional form of the response to individual variables and/or the limited nature of the interaction effects which can be accommodated. Other classes of models, such as multi-variate adaptive regression spline models of high-order (Friedman, 1991), interaction splines (Wahba, 1990) and especially non-parametric regression techniques (Hardle, 1990) are capable of relaxing some or all of these restrictive assumptions, but run the converse risk of suffering high-variance, or "over fitting".

A large literature of experimental results suggests that, under the right conditions, the flexibility of neural networks allows them to out-perform other techniques. Where the current understanding is limited, however, is in analysing trained neural networks to understand how the degrees of freedom have been allocated, in a way which allows meaningful comparisons with other classes of models. We propose that the notion of

"equivalent kernels" [eg. (Hastie and Tibshirani, 1990)] can provide a unifying framework for neural networks and other classes of regression model, as well as providing important information about the neural network itself. We describe an algorithm for estimating equivalent kernels for neural networks which overcomes the limitations of existing analytical methods.

In the following section we describe the concept of equivalent kernels. In Section 3 we describe an algorithm which estimates how the response function learned by the neural network would change if the training data were modified slightly, from which we derive the equivalent kernels for the network. Section 4 provides simulation results for two controlled experiments. Section 5 contains a brief discussion of some of the implications of this work, and highlights a number of interesting directions for further research. A summary of the main points of the paper is presented in Section 6.

## 2    EQUIVALENT KERNELS

Non-parametric regression techniques, such as kernel smoothing, local regression and nearest neighbour regression, can all be expressed in the form:

$$y(z) = \int_{x=-\infty}^{\infty} \varphi(z,x). f(x). t(x)\, dx \tag{1}$$

where $y(z)$ is the response at the query point $z$, $\varphi(z, x)$ is the weighting, or *kernel*, which is "centred" at $z$, $f(x)$ is the input density and $t(x)$ is the target function.

In finite samples, this is approximated by:

$$y(x_i) = \sum_{j=1}^{n} \varphi(x_i, x_j). t_j \tag{2}$$

and the response at point $x_i$ is a weighted average of the sampled target values across the entire dataset. Furthermore, the response can be viewed as a *least squares estimate* for $y(x_i)$ because we can write it as a solution to the minimization problem:

$$\min_{y(x_i)} \left( \sum_{j=1}^{n} \varphi(x_i, x_j). t_j - y(x_i) \right)^2 \tag{3}$$

We can combine the kernel functions to define the smoother matrix S, given by:

$$S = \begin{bmatrix} \varphi(x_1, x_1) & \varphi(x_1, x_2) & \cdots & \varphi(x_1, x_n) \\ \varphi(x_2, x_1) & \varphi(x_2, x_2) & & \vdots \\ \vdots & & \ddots & \vdots \\ \varphi(x_n, x_1) & \cdots & \cdots & \varphi(x_n, x_n) \end{bmatrix} \tag{4}$$

From which we obtain:

$$y = S.t \tag{5}$$

Where $\mathbf{y} = (\ y(x_1),\ y(x_2),\ ...\ ,\ y(x_n)\ )^T$, and $\mathbf{t} = (\ t_1,\ t_2,\ ...,\ t_n)^T$ is the vector of target values.

From the smoother matrix S, we can derive many kinds of important information. The model is represented in terms of the influence of each observation on the response at each sample point, allowing us to quantify the effect of outliers for instance. It is also possible to calculate the model bias and variance at each sample point [see (Hardle, 1990) for details]. One important measure which we will return to below is the number of degrees of freedom which are absorbed by the model; a number of definitions can be motivated, but in the case of least squares estimators they turn out to be equivalent [see pp 52-55 of (Hastie and Tibshirani, 1990)], perhaps the most intuitive is:

$$\text{dof}_S = \text{trace}(\ S\ ) \tag{6}$$

thus a model which is a look up table, i.e. $y(x_i) = t_i$, absorbs all 'n' degrees of freedom, whereas the sample mean, $y(x_i) = 1/n\ \Sigma\ t_i$ , absorbs only one degree of freedom. The degrees of freedom can be taken as a natural measure of model complexity, which formulated with respect to the data itself, rather than to the number of parameters.

The discussion above relates only to models which can be expressed in the form given by equation (2), i.e. where the "kernel functions" can be computed. Fortunately, many types of parametric models can be "inverted" in this manner, providing what are known as "equivalent kernels". Consider a model of the form:

$$y(x) = \Sigma_j\ \phi_j(x).w_j \tag{7}$$

i.e. a weighted function of some arbitrary transformations of the input variables. In the case of fitting using a least squares approach, then the optimal weights $\mathbf{w} = (\ w_1,\ w_2,\ ...,\ w_n)^T$ are given by:

$$\mathbf{w} = \Phi^+\mathbf{t} \tag{8}$$

where $\Phi^+$ is the pseudo-inverse of the transformed data matrix $\Phi$. The network output can then be expressed as:

$$y(x_i) = \Sigma_j\ \phi_j(x_i).(\ \Sigma_{k=1..n}\ [\Phi^+]_{j,k}.t_k\ ) \tag{9}$$

$$= \Sigma_k[\Sigma_j\ \phi_j(x_i)[\Phi^+]_{j,k}].t_k$$

$$= \Sigma_k\ \varphi(x_i,\ x_k).t_k$$

and the $\varphi(x_i,\ x_k)$ are then the "equivalent kernels" of the original model which is now in the same form as equation (2). Examples of equivalent kernels for different classes of parametric and non-parametric models are given by (Hastie and Tibshirani, 1990) whilst a treatment for Radial Basis Function (RBF) networks is presented in (Lowe, 1995).

## 3      EQUIVALENT KERNELS FOR NEURAL NETWORKS

The analytic approach described above relies on the ability to calculate the optimal weights using the pseudo-inverse of the data matrix. This is only possible if the transformations, $\phi(x)$, are fixed functions, as is typically the case in parametric models or single-layer neural networks. However, for a neural network with more than one layer of

adjustable weights, the basis functions are parametrised rather than fixed and are thus themselves a function of the training data. Consequently the equivalent kernels are also dependent on the data, and the problem of finding the equivalent kernels becomes non-linear.

We adopt a solution to this problem which is based on the following observation. In the case where the equivalent kernels are independent of the observed values $t_i$, we notice from equation (2):

$$\frac{\partial y_i}{\partial t_j} = \varphi(x_i, x_j) \tag{10}$$

i.e. the basis function $\varphi(x_i, x_j)$ is equal to the sensitivity of the response $y(x_i)$ to a small change in the observed value $t_j$. This suggests that we approximate the equivalent kernels by turning the above expression around:

$$\varphi(x_i, x_j) = \big(\psi(x_i) - y(x_i)\big)/\varepsilon \tag{11}$$

where $\varepsilon$ is a small perturbation of the training data and $\varphi(x_i)$ is the response of the re-optimised network:

$$\psi(x_i) = \varphi^*(x_i, x_j).(t_j + \varepsilon) + \sum_{k \neq j} \varphi^*(x_i, x_k).t_k \tag{12}$$

The notation $\varphi^*$ indicates that the new kernel functions derive from the network fitted to perturbed data. Note that this takes into account **all** of the adjustable parameters in the network. Whereas treating the basis functions as fixed would give simply the number of additive terms in the final layer of the network.

Calculating the equivalent kernels in this fashion is a computationally intensive procedure, with the network needing to be retrained after perturbing each point in turn. Note that regularisation techniques such as weight decay should be incorporated within this procedure as with initial training and are thus correctly accounted for by the algorithm. The retraining step is facilitated by using the optimised weights from the unperturbed data, causing the network to re-train from weights which are initially almost optimal (especially if the perturbation is small).

## 4    SIMULATION RESULTS

In order to investigate the practical viability of estimating equivalent kernels using the perturbation approach, we performed a controlled experiment on simulated data. The target function used was the first two periods of a sine-wave, sampled at 41 points evenly spaced between 0 and $4\pi$. This function was estimated using a neural network with a single layer of four sigmoid units, a shortcut connection from input to output, and a linear output unit, trained using standard backpropogation.

From the trained network we then estimated the equivalent kernels using the perturbation method described in the previous section. The resulting kernels for points 0, $\pi$, and $2\pi$ are shown in figure 2, below.

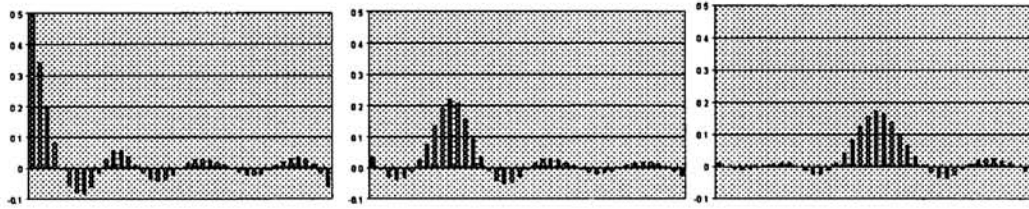

Figure 2: Equivalent Kernels for sine-wave problem

As discussed in the previous section, we can combine the estimated kernels to construct a linear smoother. The correlation coefficient between the function reconstructed from the approximated smoother matrix and the original neural network is found to be 0.995.

From equation (6) we find that the network contains approx. 8.2 degrees of freedom; this compares to the 10 *potential* degrees of freedom, and also to the 6 degrees of freedom which we would expect for an equivalent model with fixed transfer functions. Clearly, to some degreee, perturbations in the training data are accommodated by adjustments to the sigmoid functions.

Using this approach we can also investigate the effects of weight decay on (a) the ability of the network to reproduce the target function, (b) the number of degrees of freedom absorbed by the network, and (c) the kernel functions themselves. We use a standard quadratic weight decay, leading to a cost function of the form:

$$C = (y - f(x))^2 + \gamma . \Sigma w^2 \qquad\qquad (13)$$

The effect of gradually increasing the weight decay factor, $\gamma$, on both network performance and capacity is shown in figure 3(b), below:

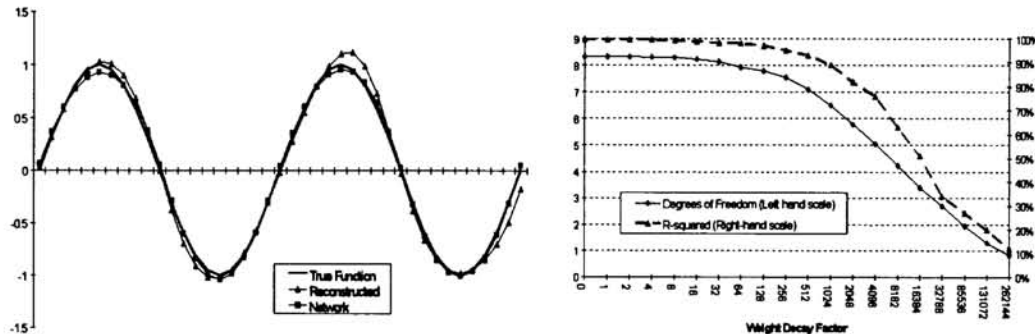

Figure 3: (a) Comparison of network and reconstructed functions with target, and (b) effect of weight decay

Looking at figure 3(b) we note that the two curves follow each other very closely. As the weight decay factor is increased, the effective capacity of the network is reduced and the performance drops off accordingly.

In one dimension, the main flexibility for the equivalent kernels is one of scale: narrow, concentrated kernels which rely heavily on nearby observations versus broad, diffuse kernels in which the response is conditioned on a larger number of observations. In higher dimensions, however, the real power of neural networks as function estimators lies in the fact that the sensitivity of the estimated network function is itself a flexible

function of the input vector. Viewed from the perspective of equivalent kernels, this property might be expected to manifest itself in a change in the *shape* of the kernels in different regions of the input space. In order to investigate this effect we applied the perturbation approach in estimating equivalent kernels for a network trained to reproduce a two-dimensional function; the function chosen was a "ring" defined by:

$$z = 1/ ( 1 + 30.( x^2 + y^2 - 0.5)^2 )$$          (14)

For ease of visualisation the input points were chosen on a regular 15 by 15 grid running between plus and minus one. This function was approximated using a 2(+1)-8-1 network with sigmoidal hidden units and a linear output unit. Selected kernel functions, estimated from this network, are shown in figure 4, below:

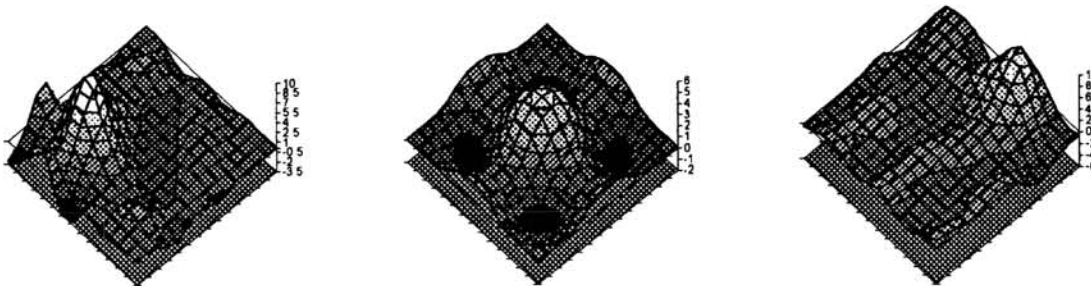

Figure 4: Equivalent Kernels: approximated using the perturbation method

This result clearly shows the changing shape of the kernel functions in different parts of the input space. The function reconstructed from the estimated smoother matrix has a correlation coefficient of 0.987 with the original network function.

## 5. Discussion

The ability to transform neural network regression models into an equivalent kernel representation raises the possibility of harnessing the whole battery of statistical methods which have been developed for non-parametric techniques: model selection procedures, prediction interval estimation, calculation of degrees of freedom, and statistical significance testing amongst others. The algorithm described in this paper raises the possibility of applying these techniques to more-powerful networks with two or more layers of adaptable weights, be they based on sigmoids, radial functions, splines or whatever, albeit at the price of significant computational effort.

Another opportunity is in the area of model combination where the added value from combining models in an *ensemble* is related to the degree of correlation between the different models (Krogh and Vedelsby, 1995). Typically the pointwise correlation between two models will be related to the similarity between their equivalent kernels and so the equivalent kernel approach opens new possibilities for conditionally modifying the ensemble weights without a need for an additional level of learning.

The influence-based method for estimating the number of degrees of freedom absorbed by a neural network model, focuses attention on uncertainty in the data itself, rather than taking the indirect route based on uncertainty in the model parameters; in future work

we propose to investigate the similarities and differences between our approach and those based on the "effective number of parameters" (Moody, 1992) and Bayesian methods (MacKay, 1992).

## 6. Summary

We suggest that equivalent kernels provide an important tool for understanding *what* neural networks do and *how* they go about doing it; in particular a large battery of existing statistical tools use information derived from the smoother matrix.

The perturbation method which we have presented overcomes the limitations of standard approaches, which are only appropriate for models with a single layer of adjustable weights, albeit at considerable computational expense. It has the added bonus of automatically taking into account the effect of regularisation techniques such as weight decay.

The experimental results illustrate the application of the technique to two simple problems. As expected the number of degrees of freedom in the models is found to be related to the amount of weight decay used during training. The equivalent kernels are found to vary significantly in different regions of input space and the functions reconstructed from the estimated smoother matrices closely match the orignal networks.

## 7. References

Breiman, L., Friedman, J. H., Olshen, R. A., and Stone C. J., 1984, *Classification and Regression Trees*, Wadsworth and Brooks/Cole, Monterey.

Friedman, J.H. and Stuetzle, W., 1981. Projection pursuit regression. *Journal of the American Statistical Association*. Vol. 76, pp. 817-823.

Friedman, J.H., 1991. Multivariate Adaptive Regression Splines (with discussion). *Annals of Statistics*. Vol 19, num. 1, pp. 1-141.

Hardle, W., 1990. *Applied nonparametric regression*. Cambridge University Press.

Hastie, T.J. and Tibshirani, R.J., 1990. *Generalised Additive Models*. Chapman and Hall, London.

Krogh, A, and Vedelsby, J., Neural network ensembles, cross-validation and active learning, *NIPS 7*, pp231-238.

Lowe, D., 1995, On the use of nonlocal and non positive definite basis functions in radial basis function networks, *Proceedings of the Fourth IEE Conference on Artificial Neural Networks*, pp. 206-211.

MacKay, D. J. C., 1992, A practical Bayesian framework for backprop networks, *Neural Computation*, 4, 448-472.

Moody, J. E., 1992, The effective number of parameters: an analysis of generalisation and regularization in nonlinear learning systems, *NIPS 4*, 847-54, Morgan Kaufmann, San Mateo

Wahba, G., 1990, *Spline Models for Observational Data*. Society for Industrial and Applied Mathematics, Philadelphia.
